# A Neural Network that Learns to Interpret Myocardial Planar Thallium Scintigrams

**Charles Rosenberg, Ph.D.***
Department of Computer Science
Hebrew University
Jerusalem, Israel

**Jacob Erel, M.D.**
Department of Cardiology
Sapir Medical Center
Meir General Hospital
Kfar Saba, Israel

**Henri Atlan, M.D., PhD.**
Department of Biophysics and Nuclear Medicine
Hadassah Medical Center
Jerusalem, Israel

## Abstract

The planar thallium-201 myocardial perfusion scintigram is a widely used diagnostic technique for detecting and estimating the risk of coronary artery disease. Neural networks learned to interpret 100 thallium scintigrams as determined by individual expert ratings. Standard error backpropagation was compared to standard LMS, and LMS combined with one layer of RBF units. Using the "leave-one-out" method, generalization was tested on all 100 cases. Training time was determined automatically from cross-validation performance. Best performance was attained by the RBF/LMS network with three hidden units per view and compares favorably with human experts.

## 1 Introduction

Coronary artery disease (CAD) is one of the leading causes of death in the Western World. The planar thallium-201 is considered to be a reliable diagnostic tool in the detection of

CAD. Thallium is a radioactive isotope that distributes in mammalian tissues after intervenous administration and is imaged by a gamma camera. The resulting scintigram is visually interpreted by the physician for the presence or absence of defects — areas with relatively lower perfusion levels. In myocardial applications, thallium is used to measure myocardial ischemia and to differentiate between viable and non-viable (infarcted) heart muscle (Pohost and Henzlova, 1990).

Diagnosis of CAD is based on the comparison of two sets of images, one set acquired immediately after a standard effort test (BRUCE protocol), and the second following a delay period of four hours. During this delay, the thallium redistributes in the heart muscle and spontaneously decays. Defects caused by scar tissue are relatively unchanged over the delay period (fixed defect), while those caused by ischemia are partially or completely filled-in (reversible defect) (Beller, 1991; Datz et al., 1992).

Image interpretation is difficult for a number of reasons: the inherent variability in biological systems which makes each case essentially unique, the vast amount of irrelevant and noisy information in an image, and the "context-dependency" of the interpretation on data from many other tests and clinical history. Interpretation can also be significantly affected by attentional shifts, perceptual abilities, and mental state (Franken Jr. and Berbaum, 1991; Cuarón et al., 1980).

While networks have found considerable application in ECG processing (e.g. (Artis et al., 1991)) and clinical decision-making (Baxt, 1991b; Baxt, 1991a), they have thus far found limited application in the field of nuclear medicine. Non-cardiac imaging applications include the grading of breast carcinomas (Dawson et al., 1991) and the discrimination of normal vs. Alzheimer's PET scans (Kippenhan et al., 1990). Of the studies dealing specifically with cardiac imaging, neural networks have been applied to several problems in cardiology including the identification of stenosis (Porenta et al., 1990; Cios et al., 1989; Cios et al., 1991; Cianflone et al., 1990; Fujita et al., 1992). These studies encouraged us to explore the use of neural networks in the interpretation of cardiac scintigraphy.

## 2    Methods

We trained one network consisting of a layer of gaussian RBF units in an unsupervised fashion to discover features in circumferential profiles in planar thallium scintigraphy. Then a second network was trained in a supervised way to map these features to physician's visual interpretations of those images using the delta rule (Widrow and Hoff, 1960). This architecture was previously found to compare favorably to other network learning algorithms (2-layer backpropagation and single-layer networks) on this task (Rosenberg et al., 1993; Erel et al., 1993).

In our experiments, all of the input vectors representing single views $\vec{I}$ were first normalized to unit length $\vec{V} = \frac{\vec{I}}{\|\vec{I}\|}$. The activation value of a gaussian unit, $O_j$, is then given by:

$$net_j = \sum_i (w_{ij} - v_i)^2 \qquad (1)$$

$$O_j = exp(-\frac{net_j}{\omega}) \qquad (2)$$

where $j$ is an index to a gaussian unit and $i$ is an input unit index. The width of the gaussian,

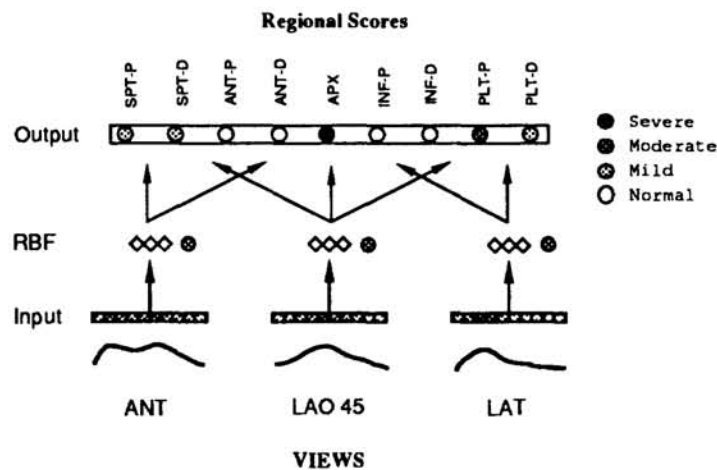

Figure 1: The network architecture. The first layer (Input) encoded the three circumferential profiles representing the three views, anterior (ANT), left lateral oblique (LAO), and left lateral (LAT). The second layer consisted of radial basis function (RBF) units, the third layer, semi-linear units trained in a supervised fashion. The outputs of the network corresponded to the visual scores as given by the expert observer. An additional unit per view encoded the scaling factor of the input patterns lost as a result of input normalization.

given by $\omega$, was fixed at 0.25 for all units[1].

The gaussian units were trained using a competitive learning rule which moves the center of the unit closest to the current input pattern ($O_{max}$, i.e. the "winner") closer to the input pattern[2]:

$$\Delta w_{i,winner} = \eta(v_i - w_{i,winner}) \qquad (3)$$

## 2.1  Data Acquisition and Selection

Scintigraphic images were acquired for each of three views: anterior (ANT), left lateral oblique (LAO 45), and left lateral (LAT) for each patient case. Acquisition was performed twice, once immediately following a standard effort test and once following a delay period of four hours. Each image was pre-processed to produce a circumferential profile (Garcia et al., 1981; Francisco et al., 1982)[3], in which maximum pixel counts within each of 60, 6° contiguous segmental regions are plotted as a function of angle (Garcia, 1991). Pre-processing involved positioning of the region of interest (ROI), interpolative background subtraction, smoothing and rotational alignment to the heart's apex (Garcia, 1991).

| Lesion | mild | moderate | severe | Total |
|--------|------|----------|--------|-------|
| single | 12 | 5 | 0 | 17 |
| multiple | 16 | 16 | 11 | 43 |
| Total | 28 | 21 | 11 | 60 |

Table 1: Distribution of Abnormal Cases as Scored by the Expert Observer. Defects occurring in any combination of two or more regions (even the proximal and distal subregions of a single area) were treated as one multiple defect. The severity level of multiple lesions was based on the most severe lesion present.

Cases were pre-selected based on the following criteria (Beller, 1991):

- **Insufficient exercise.** Cases in which the heart rate was less than 130 b.p.m. were eliminated, as this level of stress is generally deemed insufficient to accurately distinguish normal from abnormal conditions.
- **Positional abnormalities.** In a few cases, the "region of interest" was not positioned or aligned correctly by the technician.
- **Increased lung uptake.** Typically in cases of multi-vessel disease, a significant proportion of the perfusion occurs in the lungs as well as in the heart, making it more difficult to determine the condition of the heart due to the partially overlapping positions of the heart and lungs.
- **Breast artifacts.**

Cases were selected at random between August, 1989 and March, 1992. Approximately a third of the cases were eliminated due to insufficient heart rate, 4-5% due to breast artifacts, 4% due to lung uptake, and 1-2% due to positional abnormalities. A set of one hundred usable cases remained.

## 2.2 Visual Interpretation

Each case was visually scored by a single expert observer for each of nine anatomical regions generally accepted as those that best relate to the coronary circulation: *Septal: proximal* and *distal, Anterior: proximal* and *distal, Apex, Inferior: proximal* and *distal*, and *Posterior-Lateral: proximal* and *distal*. Scoring for each region was from *normal* (1) to *severe* (4), indicating the level of the observed perfusion deficit.

Intra-observer variability was examined by having the observer re-interpret 17 of the cases a second time. The observer was unable to remember the cases from the first reading and could not refer to the previous scores.

Exact matches were obtained on 91.5% of the regions; only 8 of the 153 total regions (5%) were labeled as a defect (mild, moderate or severe) on one occasion and not on the other. All differences, when they occurred, were of a single rating level[4].

## 2.3    The Network Model

The input units of the network were divided into 3 groups of 60 units each, each group representing the circumferential profile for a single view. A set of 3 RBF units were assigned to each input group. Then a second layer of weights was trained using the delta rule to reproduce the target visual scores assigned by the expert observer. The categorical visual scores were translated to numerical values to make the data suitable for network learning: *normal* = 0.0, *mild defect* = 0.3, *moderate defect* = 0.7, and *severe defect* = 1.0.

In order to make efficient use of the available data, we actually trained 100 identical networks; each network was trained on a subset of 99 of the 100 cases and tested on the remaining one. This procedure, sometimes referred to as the "leave-one-out" or "jack-knife" method, enabled us to determine the generalization performance for each case. This procedure was followed for both the RBF and the delta rule training[5]. Training of a single network took only a few minutes of Sun 4 computer time.

# 3    Results

Because of the larger numbers of confusions between normal and mild regions in both the inter- and intra-observer scores, disease was defined as moderate or severe defects. The threshold value dividing the output values of the network into these two sets was varied from 0 to 1 in 0.01 step increments. The number of agreements between the expert observer and the network were computed for each threshold value. The resulting scores, accumulated over all threshold values, were plotted as a Receiver Operating Characteristic (ROC) curve.

Best performance (percent correct) was achieved with a threshold value of 0.28, which yielded an overall accuracy of 88.7% (798/900 regions) on the stress data. However, this value of the threshold heavily favored specificity over sensitivity due to the preponderance of normal regions in the data. Using the decision threshold which maximized the sum of sensitivity and specificity, 0.10, accuracy dropped to 84.9% (764/900) but sensitivity improved to 0.771 (121/157), and specificity was 0.865 (643/743).

## 3.1    Distinguishing Fixed vs. Reversible Defects

In order to take into account the delayed distribution as well as the stress set of images, the network was essentially duplicated: one network processed the stress data, and the other,

---

ually interpreted by 3 expert observers in a previous experiment (Rosenberg et al., 1993). Percent agreement (exact matches) between the observers was 82% (288/351). Of the 63 mis-matches, 5 or about 8% of the regions were of 2 levels of severity. There were no differences of 3 levels of severity. Approximately two-thirds of the disagreements were between normal and mild regions. These results indicate that the single observer data employed in the present study are more reliable than the mixed consensus and individual scores used previously.

[5]Details of network learning were as follows: Each of the 100 networks was initialized and trained in the same way. RBF-to-output unit weights were initialized to small random values between 0.5 and -0.5. Input-to-RBF unit weights were first randomized and then normalized so that the weight vectors to each RBF unit were of unit length. Unsupervised, competitive training of the RBF units continued for 100 "epochs" or complete sweeps through the set of 99 cases: 20 epochs with a learning rate ($\eta$) of 0.1 followed by 80 epochs at 0.01 without momentum ($\alpha$). Supervised training using a learning rate of 0.05 and momentum 0.9, was terminated based on cross-validation testing after 200 epochs. Further training led to over-training and poorer generalization.

the redistribution data. (For details, see (Erel et al., 1993).)

The combined network exhibited only a limited ability to distinguish between scar and ischemia. Performance on scar detection was good (sens. 0.728 (75/103), spec. 0.878 (700/797)), but the sensitivity of the network on ischemia detection was only 0.185 (10/54). This result may be explained, at least in part, by the much smaller number of ischemic regions included in the data set as compared with scars (54 versus 103).

## 4    Conclusions and Future Directions

We suspect that our major limitation is in defect sampling. In order that a statistical system (networks or otherwise) generalize well to new cases, the data used in training must be representative of the full population of data likely to be sampled. This is unlikely to happen when the number of positive cases is on the order of 50, as was the case with ischemia, since each possible defect location, *plus* all the possible combinations of locations must be included.

A variant of backpropagation, called competitive backpropagation, has recently been developed which is claimed to generalize appropriately in the presence of multiple defects (Cho and Reggia, 1993). Weights in this network are constrained to take on positive values, so that diagnoses made by the system add constructively. In a standard backpropagation network, multiple diseases can cancel each other out, due to complex interactions of both positive and negative connection strengths. We are currently planning to investigate the application of this learning algorithm to the problem of ischemia detection.

Other improvements and extensions include:

- **Elicit confidence ratings.** Expert visual interpretations could be augmented by degree of confidence ratings. Highly ambiguous cases could be reduced in importance or eliminated. The ratings could also be used as additional targets for the network[6]: cases indicated by the network with low levels of confidence would require closer inspection by a physician. Initial results are promising in this regard.

- **Provide additional information.** We have not yet incorporated clinical history, gender, and examination EKG. Clinical history has been found to have a profound impact on interpretation of radiographs (Doubilet and Herman, 1981). The inclusion of these variables should allow the network to approximate more closely a complete diagnosis, and boost the utility of the network in the clinical setting.

- **Add constraints.** Currently we do not utilize the angles that relate the three views. It may be possible to build these angles in as constraints and thereby cut down on the number of free network parameters.

- **Expand application.** Besides planar thallium, our approach may also be applied to non-planar 3-D imaging technologies such as SPECT and other nuclear agents or stress-inducing modalities such as dipyridamole. Preliminary results are promising in this regard.

**Acknowledgements**

The authors wish to thank Mr. Haim Karger for technical assistance, and the Departments of Computer Science and Psychology at the Hebrew University for computational support. We would also like to thank Drs. David Shechter, Moshe Bocher, Roland Chisin and the staff of the Department of Medical Biophysics and Nuclear Medicine for their help, both large and small, and two anonymous reviewers. Terry Sejnowski suggested our use of RBF units.

## Footnotes

*Current address: Geriatrics, Research, Educational and Clinical Center, VA Medical Center, Salt Lake City, Utah.

[1]We have considered applying the learning rule to the unit widths ($\omega$) as well as the RBF weights, however we have not as yet pursued this possibility.

[2]Following Rumelhart and Zipser (Rumelhart and Zipser, 1986), the other units were also pulled towards the input vector, although to a much smaller extent than the winner. We used a ratio of 1 to 100.

[3]The profiles were generated using the Elscint CTL software package for planar quantitative thallium-201 based on the Cedars-Sinai technique (Garcia et al., 1981; Maddahi et al., 1981; Areeda et al., 1982).

[4]In contrast, measured *inter*-observer variability was much higher. A set of 13 cases was individ-

[6]See (Tesauro and Sejnowski, 1988) for a related idea.

# References

Areeda, J., Train, K. V., Garcia, E. V., Maddahi, J., Rosanki, A., Waxman, A., and Berman, D. (1982). Improved analysis of segmental thallium-201 myocardial scintigrams: Quantitation of distribution, washout, and redistribution. In Esser, P. D., editor, *Digital Imaging*. Society of Nuclear Medicine, New York.

Artis, S., Mark, R., and Moody, G. (1991). Detection of atrial fibrillation using artificial neural networks. In *Computers in Cardiology*, pages 173–176, Venice, Italy. IEEE, IEEE Computer Society Press.

Baxt, W. (1991a). Use of an artificial neural network for data analysis in clinical decision-making: The diagnosis of acute coronary occlusion. *Neural Computation*, 2:480–489.

Baxt, W. (1991b). Use of an artificial neural network for the diagnosis of myocardial infarction. *Annals of Internal Medicine*, 115:843–848.

Beller, G. A. (1991). Myocardial perfusion imaging with thallium-201. In Marcus, M. L., Schelbert, H. R., Skorton, D. J., and Wolf, G. L., editors, *Cardiac Imaging*. W. B. Sanders.

Cho, S. and Reggia, J. (1993). Multiple disorder diagnosis with adaptive competitive neural networks. *Artificial Intelligence in Medicine*. To appear.

Cianflone, D., Carandente, O., Fragasso, G., Margononato, A., Meloni, C., Rossetti, E., Gerundini, P., and Chiechia, S. L. (1990). A neural network based model of predicting the probability of coronary lesion from myocardial perfusion SPECT data. In *Proceedings of the 37th Annual Meeting of the Society of Nuclear Medicine*, page 797.

Cios, K. J., Goodenday, L. S., Merhi, M., and Langenderfer, R. (1989). Neural networks in detection of coronary artery disease. In *Computers in Cardiology Conference*, pages 33–37, Jerusalem, Israel. IEEE, IEEE Computer Society Press.

Cios, K. J., Shin, I., and Goodenday, L. S. (1991). Using fuzzy sets to diagnose coronary artery stenosis. *Computer*, pages 57–63.

Cuarón, A., Acero, A., Cárdena, M., Huerta, D., Rodríguez, A., and de Garay, R. (1980). Interobserver variability in the interpretation of myocardial images with Tc-99m-labeled diphosponate and pyrophosphate. *Journal of Nuclear Medicine*, 21(1):1–9.

Datz, F., Gabor, F., Christian, P., Gullber, G., Menzel, C., and Morton, K. (1992). The use of computer-assisted diagnosis in cardiac-perfusion nuclear medicine studies: A review. *Journal of Digital Imaging*, 5(4):1–14.

Dawson, A., Austin, R., and Weinberg, D. (1991). Nuclear grading of breast carcinoma by image analysis. *American Journal of Clinical Pathology*, 95(4):S29–S37.

Doubilet, P. and Herman, P. (1981). Interpretation of radiographs: Effect of clinical history. *American Journal of Roentgenology*, 137:1055–1058.

Erel, J., Rosenberg, C., and Atlan, H. (1993). Neural network for automatic interpretation of thallium scintigrams. In preparation.

Francisco, D. A., Collins, S. M., and et al., R. T. G. (1982). Tomographic thallium-201 myocardial perfusion scintigrams after maximal coronary artery vasodiliation with intravenous dipyridamole: Comparison of qualitative and quantitative approaches. *Circulation*, 66(2).

Franken Jr., E. A. and Berbaum, K. S. (1991). Perceptual aspects of cardiac imaging. In Marcus, M. L., Schelbert, H. R., Skorton, D. J., and Wolf, G. L., editors, *Cardiac Imaging*. W. B. Sanders.

Fujita, H., Katafuchi, T., Uehara, T., and Nishimura, T. (1992). Application of artificial neural network to computer-aided diagnosis of coronary artery disease in myocardial SPECT bull's-eye images. *The Journal of Nuclear Medicine*, 33(2):272–276.

Garcia, E. V. (1991). Physics and instrumentation of radionuclide imaging. In Marcus, M. L., Schelbert, H. R., Skorton, D. J., and Wolf, G. L., editors, *Cardiac Imaging*. W. B. Sanders.

Garcia, E. V., Maddahi, J., Berman, D. S., and Waxman, A. (1981). Space-time quantitation of thallium-201 myocardial scintigraphy. *Journal of Nuclear Medicine*, 22:309–317.

Kippenhan, J., Barker, W., Pascal, S., and Duara, R. (1990). A neural-network classifier applied to PET scans of normal and Alzheimer's disease (AD) patients. In *The Proceedings of the 37th Annual Meeting of the Society of Nuclear Medicine*, volume 31, Washington, D.C.

Maddahi, J., Garcia, E. V., Berman, D. S., Waxman, A., Swan, H. J. C., and Forrester, J. (1981). Improved noninvasive assessment of coronary artery disease by quantitative analysis of regional stress myocardial distribution and washout of thallium-201. *Circulation*, 64:924–935.

Pohost, G. M. and Henzlova, M. J. (1990). The value of thallium-201 imaging. *New England Journal of Medicine*, 323(3):190–192.

Porenta, G., Kundrat, S., Dorffner, G., Petta, P., Duit, J., and r, H. S. (1990). Computer based image interpretations of thallium- 201 scintigrams: Assessment of coronary artery disease using the parallel distributed processing approach. In *Proceedings of the 37th Annual Meeting of the Society of Nuclear Medicine*, page 825.

Rosenberg, C., Erel, J., and Atlan, H. (1993). A neural network that learns to interpret myocardial planar thallium scintigrams. *Neural Computation*. To appear.

Rumelhart, D. and Zipser, D. (1986). Feature discovery by competitive learning. In Rumelhart, D. and McClelland, J., editors, *Parallel Distributed Processing*, volume 1, chapter 5, pages 151–193. MIT Press, Cambridge, Mass.

Tesauro, G. and Sejnowski, T. J. (1988). A parallel network that learns to play backgammon. Technical Report CCSR-88-2, University of Illinois at Urbana-Champaign Center for Complex Systems Research.

Widrow, B. and Hoff, M. (1960). Adaptive switching circuits. In *1960 IRE WESCON Convention Record*, volume 4, pages 96–104. IRE, New York.
